# Unsupervised feature learning for audio classification using convolutional deep belief networks

**Honglak Lee**    **Yan Largman**    **Peter Pham**    **Andrew Y. Ng**
Computer Science Department
Stanford University
Stanford, CA 94305

## Abstract

In recent years, deep learning approaches have gained significant interest as a way of building hierarchical representations from unlabeled data. However, to our knowledge, these deep learning approaches have not been extensively studied for auditory data. In this paper, we apply convolutional deep belief networks to audio data and empirically evaluate them on various audio classification tasks. In the case of speech data, we show that the learned features correspond to phones/phonemes. In addition, our feature representations learned from unlabeled audio data show very good performance for multiple audio classification tasks. We hope that this paper will inspire more research on deep learning approaches applied to a wide range of audio recognition tasks.

## 1    Introduction

Understanding how to recognize complex, high-dimensional audio data is one of the greatest challenges of our time. Previous work [1, 2] revealed that learning a sparse representation of auditory signals leads to filters that closely correspond to those of neurons in early audio processing in mammals. For example, when sparse coding models are applied to natural sounds or speech, the learned representations (basis vectors) showed a striking resemblance to the cochlear filters in the auditory cortex. In related work, Grosse et al. [3] proposed an efficient sparse coding algorithm for auditory signals and demonstrated its usefulness in audio classification tasks.

However, the proposed methods have been applied to learn relatively shallow, one-layer representations. Learning more complex, higher-level representation is still a non-trivial, challenging problem. Recently, many promising approaches have been proposed to learn the processing steps of the "second stage and beyond" [4, 5, 6, 7, 8]. These "deep learning" algorithms try to learn simple features in the lower layers and more complex features in the higher layers. However, to the best of our knowledge, these "deep learning" approaches have not been extensively applied to auditory data.

The deep belief network [4] is a generative probabilistic model composed of one visible (observed) layer and many hidden layers. Each hidden layer unit learns a statistical relationship between the units in the lower layer; the higher layer representations tend to become more complex. The deep belief network can be efficiently trained using greedy layerwise training, in which the hidden layers are trained one at a time in a bottom-up fashion [4]. Recently, convolutional deep belief networks [9] have been developed to scale up the algorithm to high-dimensional data. Similar to deep belief networks, convolutional deep belief networks can be trained in a greedy, bottom-up fashion. By applying these networks to images, Lee et al. (2009) showed good performance in several visual recognition tasks [9].

In this paper, we will apply convolutional deep belief networks to unlabeled auditory data (such as speech and music) and evaluate the learned feature representations on several audio classification tasks. In the case of speech data, we show that the learned features correspond to phones/phonemes. In addition, our feature representations outperform other baseline features (spectrogram and MFCC)

for multiple audio classification tasks. In particular, our method compares favorably with other state-of-the-art algorithms for the speaker identification task. For the phone classification task, MFCC features can be augmented with our features to improve accuracy. We also show for certain tasks that the second-layer features produce higher accuracy than the first-layer features, which justifies the use of deep learning approaches for audio classification. Finally, we show that our features give better performance in comparison to other baseline features for music classification tasks. In our experiments, the learned features often performed much better than other baseline features when there was only a small number of labeled training examples. To the best of our knowledge, we are the first to apply deep learning algorithms to a range of audio classification tasks. We hope that this paper will inspire more research on deep learning approaches applied to audio recognition tasks.

## 2 Algorithms

### 2.1 Convolutional deep belief networks

We first briefly review convolutional restricted Boltzmann machines (CRBMs) [9, 10, 11] as building blocks for convolutional deep belief networks (CDBNs). We will follow the formulation of [9] and adapt it to a one dimensional setting. For the purpose of this explanation, we assume that all inputs to the algorithm are single-channel time-series data with $n_V$ frames (an $n_V$ dimensional vector); however, the formulation can be straightforwardly extended to the case of multiple channels.

The CRBM is an extension of the "regular" RBM [4] to a convolutional setting, in which the weights between the hidden units and the visible units are shared among all locations in the hidden layer. The CRBM consists of two layers: an input (visible) layer $V$ and a hidden layer $H$. The hidden units are binary-valued, and the visible units are binary-valued or real-valued.

Consider the input layer consisting of an $n_V$ dimensional array of binary units. To construct the hidden layer, consider $K$ $n_W$-dimensional filter weights $W^K$ (also referred to as "bases" throughout this paper). The hidden layer consists of $K$ "groups" of $n_H$-dimensional arrays (where $n_H \triangleq n_V - n_W + 1$) with units in group $k$ sharing the weights $W^k$. There is also a shared bias $b_k$ for each group and a shared bias $c$ for the visible units. The energy function can then be defined as:

$$E(\mathbf{v}, \mathbf{h}) = -\sum_{k=1}^{K}\sum_{j=1}^{n_H}\sum_{r=1}^{n_W} h_j^k W_r^k v_{j+r-1} - \sum_{k=1}^{K} b_k \sum_{j=1}^{n_H} h_j^k - c\sum_{i=1}^{n_V} v_i. \tag{1}$$

Similarly, the energy function of CRBM with real-valued visible units can be defined as:

$$E(\mathbf{v}, \mathbf{h}) = \frac{1}{2}\sum_{i}^{n_V} v_i^2 - \sum_{k=1}^{K}\sum_{j=1}^{n_H}\sum_{r=1}^{n_W} h_j^k W_r^k v_{j+r-1} - \sum_{k=1}^{K} b_k \sum_{j=1}^{n_H} h_j^k - c\sum_{i=1}^{n_V} v_i. \tag{2}$$

The joint and conditional probability distributions are defined as follows:

$$P(\mathbf{v}, \mathbf{h}) = \frac{1}{Z}\exp(-E(\mathbf{v}, \mathbf{h})) \tag{3}$$

$$P(h_j^k = 1|\mathbf{v}) = sigmoid((\tilde{W}^k *_v v)_j + b_k) \tag{4}$$

$$P(v_i = 1|\mathbf{h}) = sigmoid(\sum_k (W^k *_f h^k)_i + c) \quad \text{(for binary visible units)} \tag{5}$$

$$P(v_i|\mathbf{h}) = Normal(\sum_k (W^k *_f h^k)_i + c, 1) \quad \text{(for real visible units)}, \tag{6}$$

where $*_v$ is a "valid" convolution, $*_f$ is a "full" convolution,[1] and $\tilde{W}_j^k \triangleq W_{n_W - j+1}^k$. Since all units in one layer are conditionally independent given the other layer, inference in the network can be efficiently performed using block Gibbs sampling. Lee et al. [9] further developed a convolutional RBM with "probabilistic max-pooling," where the maxima over small neighborhoods of hidden units are computed in a probabilistically sound way. (See [9] for more details.) In this paper, we use CRBMs with probabilistic max-pooling as building blocks for convolutional deep belief networks.

For training the convolutional RBMs, computing the exact gradient for the log-likelihood term is intractable. However, contrastive divergence [12] can be used to approximate the gradient effectively. Since a typical CRBM is highly overcomplete, a sparsity penalty term is added to the log-likelihood objective [8, 9]. More specifically, the training objective can be written as

$$\text{minimize}_{W,b,c} \qquad \mathcal{L}_{likelihood}(W, b, c) + \mathcal{L}_{sparsity}(W, b, c), \qquad (7)$$

where $\mathcal{L}_{likelihood}$ is a negative log-likelihood that measures how well the CRBM approximates the input data distribution, and $\mathcal{L}_{sparsity}$ is a penalty term that constrains the hidden units to having sparse average activations. This sparsity regularization can be viewed as limiting the "capacity" of the network, and it often results in more easily interpretable feature representations. Once the parameters for all the layers are trained, we stack the CRBMs to form a convolutional deep belief network. For inference, we use feed-forward approximation.

## 2.2 Application to audio data

For the application of CDBNs to audio data, we first convert time-domain signals into spectrograms. However, the dimensionality of the spectrograms is large (e.g., 160 channels). We apply PCA whitening to the spectrograms and create lower dimensional representations. Thus, the data we feed into the CDBN consists of $n_c$ channels of one-dimensional vectors of length $n_V$, where $n_c$ is the number of PCA components in our representation. Similarly, the first-layer bases are comprised of $n_c$ channels of one-dimensional filters of length $n_W$.

# 3 Unsupervised feature learning

## 3.1 Training on unlabeled TIMIT data

We trained the first and second-layer CDBN representations using a large, unlabeled speech dataset. First, we extracted the spectrogram from each utterance of the TIMIT training data [13]. The spectrogram had a 20 ms window size with 10 ms overlaps. The spectrogram was further processed using PCA whitening (with 80 components) to reduce the dimensionality. We then trained 300 first-layer bases with a filter length ($n_W$) of 6 and a max-pooling ratio (local neighborhood size) of 3. We further trained 300 second-layer bases using the max-pooled first-layer activations as input, again with a filter length of 6 and a max-pooling ratio of 3.

## 3.2 Visualization

In this section, we illustrate what the network "learns" through visualization. We visualize the first-layer bases by multiplying the inverse of the PCA whitening on each first-layer basis (Figure 1). Each second-layer basis is visualized as a weighted linear combination of the first-layer bases.

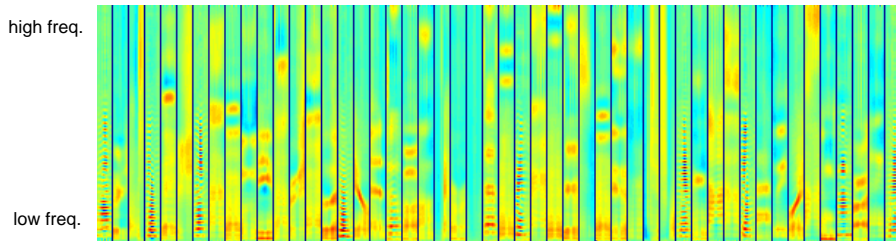

Figure 1: Visualization of randomly selected first-layer CDBN bases trained on the TIMIT data. Each column represents a "temporal receptive field" of a first-layer basis in the spectrogram space. The frequency channels are ordered from the lowest frequency (bottom) to the highest frequency (top). All figures in the paper are best viewed in color.

### 3.2.1 Phonemes and the CDBN features

In Figure 2, we show how our bases relate to phonemes by comparing visualizations of each phoneme with the bases that are most activated by that phoneme.

For each phoneme, we show five spectrograms of sound clips of that phoneme (top five columns in each phoneme group), and the five first-layer bases with the highest average activations on the given phoneme (bottom five columns in each phoneme group). Many of the first-layer bases closely match the shapes of phonemes. There are prominent horizontal bands in the lower frequencies of the first-layer bases that respond most to vowels (for example, "ah" and "oy"). The bases that respond most

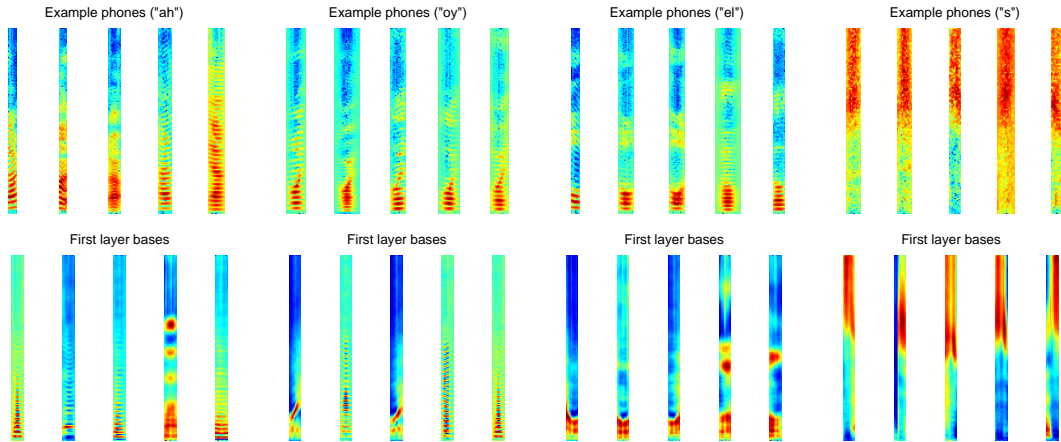

Figure 2: Visualization of the four different phonemes and their corresponding first-layer CDBN bases. For each phoneme: (top) the spectrograms of the five randomly selected phones; (bottom) five first-layer bases with the highest average activations on the given phoneme.

to fricatives (for example, "s") typically take the form of widely distributed areas of energy in the high frequencies of the spectrogram. Both of these patterns reflect the structure of the corresponding phoneme spectrograms.

Closer inspection of the bases provides slight evidence that the first-layer bases also capture more fine-grained details. For example, the first and third "oy" bases reflect the upward-slanting pattern in the phoneme spectrograms. The top "el" bases mirror the intensity patterns of the corresponding phoneme spectrograms: a high intensity region appears in the lowest frequencies, and another region of lesser intensity appears a bit higher up.

### 3.2.2 Speaker gender information and the CDBN features

In Figure 3, we show an analysis of two-layer CDBN feature representations with respect to the gender classification task (Section 4.2). Note that the network was trained on unlabeled data; therefore, no information about speaker gender was given during training.

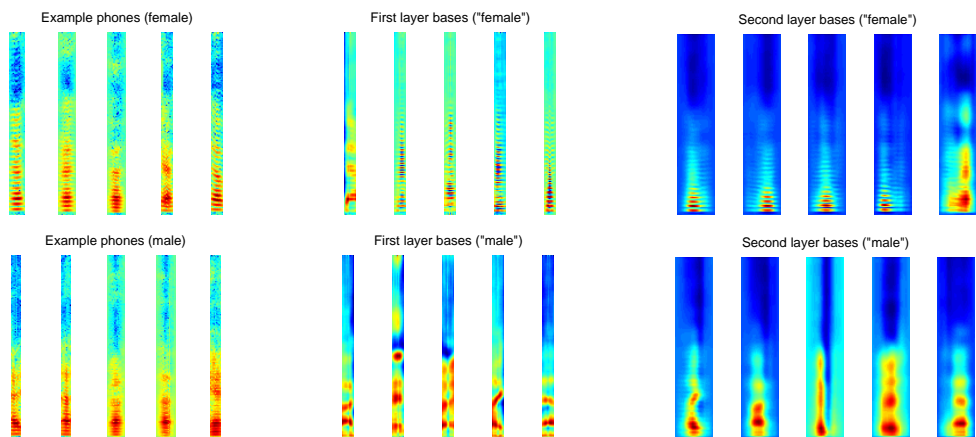

Figure 3: (Left) five spectrogram samples of "ae" phoneme from female (top)/male (bottom) speakers. (Middle) Visualization of the five first-layer bases that most differentially activate for female/male speakers. (Right) Visualization of the five second-layer bases that most differentially activate for female/male speakers.

For comparison with the CDBN features, randomly selected spectrograms of female (top left five columns) and male (bottom left five columns) pronunciations of the "ae" phoneme from the TIMIT dataset are shown. Spectrograms for the female pronunciations are qualitatively distinguishable by a finer horizontal banding pattern in low frequencies, whereas male pronunciations have more blurred

patterns. This gender difference in the vowel pronunciation patterns is typical across the TIMIT data.

Only the bases that are most biased to activate on either male or female speech are shown. The bases that are most active on female speech encode the horizontal band pattern that is prominent in the spectrograms of female pronunciations. On the other hand, the male-biased bases have more blurred patterns, which again visually matches the corresponding spectrograms.

## 4 Application to speech recognition tasks

In this section, we demonstrate that the CDBN feature representations learned from the unlabeled speech corpus can be useful for multiple speech recognition tasks, such as speaker identification, gender classification, and phone classification. In most of our experiments, we followed the self-taught learning framework [14]. The motivation for self-taught learning comes from situations where we are given only a small amount of labeled data and a large amount of unlabeled data;[2] therefore, one of our main interests was to evaluate the different feature representations given a small number of labeled training examples (as often assumed in self-taught learning or semi-supervised learning settings). More specifically, we trained the CDBN on unlabeled TIMIT data (as described in Section 3.1); then we used the CDBN features for classification on labeled training/test data[3] that were randomly selected from the TIMIT corpus.[4]

### 4.1 Speaker identification

We evaluated the usefulness of the learned CDBN representations for the speaker identification task. The subset of the TIMIT corpus that we used for speaker identification has 168 speakers and 10 utterances (sentences) per speaker, resulting in a total of 1680 utterances. We performed 168-way classification on this set. For each number of utterances per speaker, we randomly selected training utterances and testing utterances and measured the classification accuracy; we report the results averaged over 10 random trials.[5] To construct training and test data for the classification task, we extracted a spectrogram from each utterance in the TIMIT corpus. We denote this spectrogram representation as "RAW" features. We computed the first and second-layer CDBN features using the spectrogram as input. We also computed MFCC features, widely-used standard features for generic speech recognition tasks. As a result, we obtained spectrogram/MFCC/CDBN representations for each utterance with multiple (typically, several hundred) frames. In our experiments, we used simple summary statistics (for each channel) such as average, max, or standard deviation over all the frames. We evaluated the features using standard supervised classifiers, such as SVM, GDA, and KNN. The choices of summary statistics and hyperparameters for the classifiers were done using cross-validation. We report the average classification accuracy (over 10 random trials) with a varying number of training examples.

Table 1 shows the average classification accuracy for each feature representation. The results show that the first and second CDBN representations both outperform baseline features (RAW and MFCC). The numbers compare MFCC and CDBN features with as many of the same factors (such as preprocessing and classification algorithms) as possible. Further, to make a fair comparison between CDBN features and MFCC, we used the best performing implementation[6] among several standard implementations for MFCC. Our results suggest that without special preprocessing or postprocess-

Table 1: Test classification accuracy for speaker identification using summary statistics

| #training utterances per speaker | RAW | MFCC | CDBN L1 | CDBN L2 | CDBN L1+L2 |
|---|---|---|---|---|---|
| 1 | 46.7% | 54.4% | **74.5%** | 62.8% | 72.8% |
| 2 | 43.5% | 69.9% | **76.7%** | 66.2% | **76.7%** |
| 3 | 67.9% | 76.5% | 91.3% | 84.3% | **91.8%** |
| 5 | 80.6% | 82.6% | 93.7% | 89.6% | **93.8%** |
| 8 | 90.4% | 92.0% | **97.9%** | 95.2% | 97.0% |

Table 2: Test classification accuracy for speaker identification using all frames

| #training utterances per speaker | MFCC ([16]'s method) | CDBN | MFCC ([16]) + CDBN |
|---|---|---|---|
| 1 | 40.2% | 90.0% | **90.7%** |
| 2 | 87.9% | 97.9% | **98.7%** |
| 3 | 95.9% | 98.7% | **99.2%** |
| 5 | 99.2% | 99.2% | **99.6%** |
| 8 | 99.7% | 99.7% | **100.0%** |

ing (besides the summary statistics which were needed to reduce the number of features), the CDBN features outperform MFCC features, especially in a setting with a very limited number of labeled examples.

We further experimented to determine if the CDBN features can achieve competitive performance in comparison to other more sophisticated, state-of-the-art methods. For each feature representation, we used the classifier that achieved the highest performance. More specifically, for the MFCC features we replicated Reynolds (1995)'s method,[7] and for the CDBN features we used a SVM based ensemble method.[8] As shown in Table 2, the CDBN features consistently outperformed MFCC features when the number of training examples was small. We also combined both methods by taking a linear combination of the two classifier outputs (before taking the final classification prediction from each algorithm).[9] The resulting combined classifier performed the best, achieving 100% accuracy for the case of 8 training utterances per speaker.

## 4.2   Speaker gender classification

We also evaluated the same CDBN features which were learned for the speaker identification task on the gender classification task. We report the classification accuracy for various quantities of training examples (utterances) per gender. For each number of training examples, we randomly sampled training examples and 200 testing examples; we report the test classification accuracy averaged over 20 trials. As shown in Table 3, both the first and second CDBN features outperformed the baseline features, especially when the number of training examples were small. The second-layer CDBN features consistently performed better than the first-layer CDBN features. This suggests that the second-layer representation learned more invariant features that are relevant for speaker gender classification, justifying the use of "deep" architectures.

## 4.3   Phone classification

Finally, we evaluated our learned representation on phone classification tasks. For this experiment, we treated each phone segment as an individual example and computed the spectrogram (RAW) and MFCC features for each phone segment. Similarly, we computed the first-layer CDBN representations. Following the standard protocol [15], we report the 39 way phone classification accuracy on the test data (TIMIT core test set) for various numbers of training sentences. For each number of training examples, we report the average classification accuracy over 5 random trials. The summary

Table 3: Test accuracy for gender classification problem

| #training utterances per gender | RAW | MFCC | CDBN L1 | CDBN L2 | CDBN L1+L2 |
|---|---|---|---|---|---|
| 1 | 68.4% | 58.5% | 78.5% | **85.8%** | 83.6% |
| 2 | 76.7% | 78.7% | 86.0% | **92.5%** | 92.3% |
| 3 | 79.5% | 84.1% | 88.9% | **94.2%** | **94.2%** |
| 5 | 84.4% | 86.9% | 93.1% | **95.8%** | 95.6% |
| 7 | 89.2% | 89.0% | 94.2% | **96.6%** | 96.5% |
| 10 | 91.3% | 89.8% | 94.7% | **96.7%** | 96.6% |

Table 4: Test accuracy for phone classification problem

| #training utterances | RAW | MFCC | MFCC ([15]'s method) | CDBN L1 | MFCC+CDBN L1 ([15]) |
|---|---|---|---|---|---|
| 100 | 36.9% | 58.3% | 66.6% | 53.7% | **67.2%** |
| 200 | 37.8% | 61.5% | 70.3% | 56.7% | **71.0%** |
| 500 | 38.7% | 64.9% | 74.1% | 59.7% | **75.1%** |
| 1000 | 39.0% | 67.2% | 76.3% | 61.6% | **77.1%** |
| 2000 | 39.2% | 69.2% | 78.4% | 63.1% | **79.2%** |
| 3696 | 39.4% | 70.8% | 79.6% | 64.4% | **80.3%** |

results are shown in Table 4. In this experiment, the first-layer CDBN features performed better than spectrogram features, but they did not outperform the MFCC features. However, by combining MFCC features and CDBN features, we could achieve about 0.7% accuracy improvement consistently over all the numbers of training utterances. In the realm of phone classification, in which significant research effort is often needed to achieve even improvements well under a percent, this is a significant improvement. [17, 18, 19, 20]

This suggests that the first-layer CDBN features learned somewhat informative features for phone classification tasks in an unsupervised way. In contrast to the gender classification task, the second-layer CDBN features did not offer much improvement over the first-layer CDBN features. This result is not unexpected considering the fact that the time-scale of most phonemes roughly corresponds to the time-scale of the first-layer CDBN features.

## 5 Application to music classification tasks

In this section, we assess the applicability of CDBN features to various music classification tasks.

Table 5: Test accuracy for 5-way music genre classification

| Train examples | RAW | MFCC | CDBN L1 | CDBN L2 | CDBN L1+L2 |
|---|---|---|---|---|---|
| 1 | 51.6% | 54.0% | **66.1%** | 62.5% | 64.3% |
| 2 | 57.0% | 62.1% | **69.7%** | 67.9% | 69.5% |
| 3 | 59.7% | 65.3% | **70.0%** | 66.7% | 69.5% |
| 5 | 65.8% | 68.3% | **73.1%** | 69.2% | 72.7% |

### 5.1 Music genre classification

For the task of music genre classification, we trained the first and second-layer CDBN representations on an unlabeled collection of music data.[10] First, we computed the spectrogram (20 ms window size with 10 ms overlaps) representation for individual songs. The spectrogram was PCA-whitened and then fed into the CDBN as input data. We trained 300 first-layer bases with a filter length of 10 and a max-pooling ratio of 3. In addition, we trained 300 second-layer bases with a filter length of 10 and a max-pooling ratio of 3.

We evaluated the learned CDBN representation for 5-way genre classification tasks. The training and test songs for the classification tasks were randomly sampled from 5 genres (classical, electric, jazz, pop, and rock) and did not overlap with the unlabeled data. We randomly sampled 3-second segments from each song and treated each segment as an individual training or testing example. We report the classification accuracy for various numbers of training examples. For each number of training examples, we averaged over 20 random trials. The results are shown in Table 5. In this task, the first-layer CDBN features performed the best overall.

## 5.2 Music artist classification

Furthermore, we evaluated whether the CDBN features are useful in identifying individual artists.[11] Following the same procedure as in Section 5.1, we trained the first and second-layer CDBN representations from an unlabeled collection of classical music data. Some representative bases are shown in Figure 4. Then we evaluated the learned CDBN representation for 4-way artist identification tasks. The disjoint sets of training and test songs for the classification tasks were randomly sampled from the songs of four artists. The unlabeled data and the labeled data did not include the same artists. We randomly sampled 3-second segments from each song and treated each segment as an individual example. We report the classification accuracy for various quantities of training examples. For each number of training examples, we averaged over 20 random trials. The results are shown in Table 6. The results show that both the first and second-layer CDBN features performed better than the baseline features, and that either using the second-layer features only or combining the first and the second-layer features yielded the best results. This suggests that the second-layer CDBN representation might have captured somewhat useful, higher-level features than the first-layer CDBN representation.

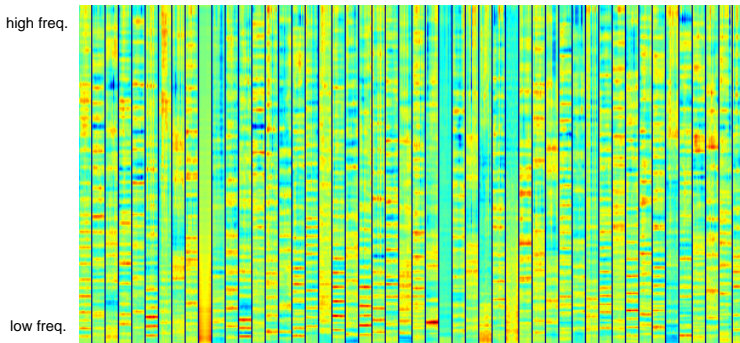

Figure 4: Visualization of randomly selected first-layer CDBN bases trained on classical music data.

Table 6: Test accuracy for 4-way artist identification

| Train examples | RAW | MFCC | CDBN L1 | CDBN L2 | CDBN L1+L2 |
|---|---|---|---|---|---|
| 1 | 56.0% | 63.7% | 67.6% | 67.7% | **69.2%** |
| 2 | 69.4% | 66.1% | 76.1% | 74.2% | **76.3%** |
| 3 | 73.9% | 67.9% | 78.0% | 75.8% | **78.7%** |
| 5 | 79.4% | 71.6% | 80.9% | **81.9%** | 81.4% |

## 6 Discussion and conclusion

Modern speech datasets are much larger than the TIMIT dataset. While the challenge of larger datasets often lies in considering harder tasks, our objective in using the TIMIT data was to restrict the amount of labeled data our algorithm had to learn from. It remains an interesting problem to apply deep learning to larger datasets and more challenging tasks.

In this paper, we applied convolutional deep belief networks to audio data and evaluated on various audio classification tasks. By leveraging a large amount of unlabeled data, our learned features often equaled or surpassed MFCC features, which are hand-tailored to audio data. Furthermore, even when our features did not outperform MFCC, we could achieve higher classification accuracy by combining both. Also, our results show that a single CDBN feature representation can achieve high performance on multiple audio recognition tasks. We hope that our approach will inspire more research on automatically learning deep feature hierarchies for audio data.

**Acknowledgment**

We thank Yoshua Bengio, Dan Jurafsky, Yun-Hsuan Sung, Pedro Moreno, Roger Grosse for helpful discussions. We also thank anonymous reviewers for their constructive comments. This work was supported in part by the National Science Foundation under grant EFRI-0835878, and in part by the Office of Naval Research under MURI N000140710747.

## Footnotes

[1]Given an $m$-dimensional vector and an $n$-dimensional kernel (where $m > n$), valid convolution gives a $(m - n + 1)$-dimensional vector, and full convolution gives a $(m + n - 1)$-dimensional vector.

[2]In self-taught learning, the labeled data and unlabeled data don't need to share the same labels or the same generative distributions.

[3]There are two disjoint TIMIT data sets. We drew unlabeled data from the larger of the two for unsupervised feature learning, and we drew labeled data from the other data set to create our training and test set for the classification tasks.

[4]In the case of phone classification, we followed the standard protocol (e.g., [15]) rather than self-taught learning framework to evaluate our algorithm in comparison to other methods.

[5]Details: There were some exceptions to this; for the case of eight training utterances, we followed Reynolds (1995) [16]; more specifically, we used eight training utterances (2 sa sentences, 3 si sentences and first 3 sx sentences); the two testing utterances were the remaining 2 sx sentences. We used cross validation for selecting hyperparameters for classification, except for the case of 1 utterance per speaker, where we used a randomly selected validation sentence per speaker.

[6]We used Dan Ellis' implementation available at: `http://labrosa.ee.columbia.edu/matlab/rastamat`.

[7]Details: In [16], MFCC features (with multiple frames) were computed for each utterance; then a Gaussian mixture model was trained for each speaker (treating each individual MFCC frame as a input example to the GMM. For the a given test utterance, the prediction was made by determining the GMM model that had the highest test log-likelihood.

[8]In detail, we treated each single-frame CDBN features as an individual example. Then, we trained a multi-class linear SVM for these individual frames. For testing, we computed SVM prediction score for each speaker, and then aggregated predictions from all the frames. Overall, the highest scoring speaker was selected for the prediction.

[9]The constant for the linear combination was fixed across all the numbers of training utterances, and it was selected using cross validation.

[10]Available from `http://ismir2004.ismir.net/ISMIR_Contest.html`.

[11]In our experiments, we found that artist identification task was more difficult than the speaker identification task because the local sound patterns can be highly variable even for the same artist.

# References

[1] E. C. Smith and M. S. Lewicki. Efficient auditory coding. *Nature*, 439:978–982, 2006.

[2] B. A. Olshausen and D. J. Field. Emergence of simple-cell receptive field properties by learning a sparse code for natural images. *Nature*, 381:607–609, 1996.

[3] R. Grosse, R. Raina, H. Kwong, and A.Y. Ng. Shift-invariant sparse coding for audio classification. In *UAI*, 2007.

[4] G. E. Hinton, S. Osindero, and Y.-W. Teh. A fast learning algorithm for deep belief nets. *Neural Computation*, 18(7):1527–1554, 2006.

[5] M. Ranzato, C. Poultney, S. Chopra, and Y. LeCun. Efficient learning of sparse representations with an energy-based model. In *NIPS*, 2006.

[6] Y. Bengio, P. Lamblin, D. Popovici, and H. Larochelle. Greedy layer-wise training of deep networks. In *NIPS*, 2006.

[7] H. Larochelle, D. Erhan, A. Courville, J. Bergstra, and Y. Bengio. An empirical evaluation of deep architectures on problems with many factors of variation. In *ICML*, 2007.

[8] H. Lee, C. Ekanadham, and A. Y. Ng. Sparse deep belief network model for visual area V2. In *NIPS*, 2008.

[9] H. Lee, R. Grosse, R. Ranganath, and A. Y. Ng. Convolutional deep belief networks for scalable unsupervised learning of hierarchical representations. In *ICML*, 2009.

[10] G. Desjardins and Y. Bengio. Empirical evaluation of convolutional RBMs for vision. Technical report, 2008.

[11] M. Norouzi, M. Ranjbar, and G. Mori. Stacks of convolutional restricted boltzmann machines for shift-invariant feature learning. In *CVPR*, 2009.

[12] G. E. Hinton. Training products of experts by minimizing contrastive divergence. *Neural Computation*, 14:1771–1800, 2002.

[13] W. Fisher, G. Doddington, and K. Goudie-Marshall. The darpa speech recognition research database: Specifications and status. In *DARPA Speech Recognition Workshop*, 1986.

[14] R. Raina, A. Battle, H. Lee, B. Packer, and A. Y. Ng. Self-taught learning: Transfer learning from unlabeled data. In *ICML*, 2007.

[15] P. Clarkson and P. J. Moreno. On the use of support vector machines for phonetic classification. In *ICASSP99*, pages 585–588, 1999.

[16] D. A. Reynolds. Speaker identification and verification using gaussian mixture speaker models. *Speech Commun.*, 17(1-2):91–108, 1995.

[17] F. Sha and L. K. Saul. Large margin gaussian mixture modeling for phonetic classication and recognition. In *ICASSP'06*, 2006.

[18] Y.-H. Sung, C. Boulis, C. Manning, and D. Jurafsky. Regularization, adaptation, and non-independent features improve hidden conditional random fields for phone classification. In *IEEE ASRU*, 2007.

[19] S. Petrov, A. Pauls, and D. Klein. Learning structured models for phone recognition. In *EMNLP-CoNLL*, 2007.

[20] D. Yu, L. Deng, and A. Acero. Hidden conditional random field with distribution constraints for phone classification. In *Interspeech*, 2009.

